# Structural Risk Minimization
# for Character Recognition

**I. Guyon, V. Vapnik, B. Boser, L. Bottou, and S. A. Solla**
AT&T Bell Laboratories
Holmdel, NJ 07733, USA

## Abstract

The method of *Structural Risk Minimization* refers to tuning the capacity of the classifier to the available amount of training data. This capacity is influenced by several factors, including: (1) properties of the input space, (2) nature and structure of the classifier, and (3) learning algorithm. Actions based on these three factors are combined here to control the capacity of linear classifiers and improve generalization on the problem of handwritten digit recognition.

## 1  RISK MINIMIZATION AND CAPACITY

### 1.1  EMPIRICAL RISK MINIMIZATION

A common way of training a given classifier is to adjust the parameters $\mathbf{w}$ in the classification function $F(\mathbf{x}, \mathbf{w})$ to minimize the *training error* $E_{train}$, i.e. the frequency of errors on a set of $p$ training examples. $E_{train}$ estimates the expected risk based on the empirical data provided by the $p$ available examples. The method is thus called *Empirical Risk Minimization*. But the classification function $F(\mathbf{x}, \mathbf{w}^*)$ which minimizes the empirical risk does not necessarily minimize the *generalization error*, i.e. the expected value of the risk over the full distribution of possible inputs and their corresponding outputs. Such generalization error $E_{gene}$ cannot in general be computed, but it can be estimated on a separate test set ($E_{test}$). Other ways of

estimating $E_{gene}$ include the *leave-one-out* or *moving control* method [Vap82] (for a review, see [Moo92]).

## 1.2 CAPACITY AND GUARANTEED RISK

Any family of classification functions $\{F(\mathbf{x}, \mathbf{w})\}$ can be characterized by its capacity. The Vapnik-Chervonenkis dimension (or VC-dimension) [Vap82] is such a capacity, defined as the maximum number $h$ of training examples which can be learnt without error, *for all possible binary labelings*. The VC-dimension is in some cases simply given by the number of free parameters of the classifier, but in most practical cases it is quite difficult to determine it analytically.

The VC-theory provides bounds. Let $\{F(\mathbf{x}, \mathbf{w})\}$ be a set of classification functions of capacity $h$. With probability $(1 - \eta)$, for a number of training examples $p > h$, simultaneously for all classification functions $F(\mathbf{x}, \mathbf{w})$, the generalization error $E_{gene}$ is lower than a *guaranteed risk* defined by:

$$E_{guarant} = E_{train} + \epsilon(p, h, E_{train}, \eta) \, , \qquad (1)$$

where $\epsilon(p, h, E_{train}, \eta)$ is proportional to $\epsilon_0 = [h(ln2p/h + 1) - \eta]/p$ for small $E_{train}$, and to $\sqrt{\epsilon_0}$ for $E_{train}$ close to one  [Vap82,Vap92].

For a fixed number of training examples $p$, the training error decreases monotonically as the capacity $h$ increases, while both guaranteed risk and generalization error go through a minimum. Before the minimum, the problem is *overdetermined*: the capacity is too small for the amount of training data. Beyond the minimum the problem is *underdetermined*. The key issue is therefore to match the capacity of the classifier to the amount of training data in order to get best generalization performance. The method of *Structural Risk Minimization* (SRM) [Vap82,Vap92] provides a way of achieving this goal.

## 1.3 STRUCTURAL RISK MINIMIZATION

Let us choose a family of classifiers $\{F(\mathbf{x}, \mathbf{w})\}$, and define a structure consisting of nested subsets of elements of the family: $S_1 \subset S_2 \subset ... \subset S_r \subset ....$ By defining such a structure, we ensure that the capacity $h_r$ of the subset of classifiers $S_r$ is less than $h_{r+1}$ of subset $S_{r+1}$. The method of SRM amounts to finding the subset $S^{opt}$ for which the classifier $F(\mathbf{x}, \mathbf{w}^*)$ which minimizes the empirical risk within such subset yields the best overall generalization performance.

Two problems arise in implementing SRM: (I) How to select $S^{opt}$? (II) How to find a good structure? Problem (I) arises because we have no direct access to $E_{gene}$. In our experiments, we will use the minimum of either $E_{test}$ or $E_{guarant}$ to select $S^{opt}$, and show that these two minima are very close. A good structure reflects the *a priori* knowledge of the designer, and only few guidelines can be provided from the theory to solve problem (II). The designer must find the best compromise between two competing terms: $E_{train}$ and $\epsilon$. Reducing $h$ causes $\epsilon$ to decrease, but $E_{train}$ to increase. A good structure should be such that decreasing the VC-dimension happens at the expense of the smallest possible increase in training error. We now examine several ways in which such a structure can be built.

# 2   PRINCIPAL COMPONENT ANALYSIS, OPTIMAL BRAIN DAMAGE, AND WEIGHT DECAY

Consider three apparently different methods of improving generalization performance: Principal Component Analysis (a preprocessing transformation of input space) [The89], Optimal Brain Damage (an architectural modification through weight pruning) [LDS90], and a regularization method, Weight Decay (a modification of the learning algorithm) [Vap82]. For the case of a linear classifier, these three approaches are shown here to control the capacity of the learning system through the same underlying mechanism: a reduction of the *effective dimension* of weight space, based on the curvature properties of the Mean Squared Error ($MSE$) cost function used for training.

## 2.1   LINEAR CLASSIFIER AND $MSE$ TRAINING

Consider a binary linear classifier $F(\mathbf{x}, \mathbf{w}) = \theta_0(\mathbf{w}^T \mathbf{x})$, where $\mathbf{w}^T$ is the transpose of $\mathbf{w}$ and the function $\theta_0$ takes two values 0 and 1 indicating to which class $\mathbf{x}$ belongs. The VC-dimension of such classifier is equal to the dimension of input space [1] (or the number of weights): $h = dim(\mathbf{w}) = dim(\mathbf{x}) = n$.

The empirical risk is given by:

$$E_{train} = \frac{1}{p} \sum_{k=1}^{p} (y^k - \theta_0(\mathbf{w}^T \mathbf{x}^k))^2 \, , \qquad (2)$$

where $\mathbf{x}^k$ is the $k^{th}$ example, and $y^k$ is the corresponding desired output. The problem of minimizing $E_{train}$ as a function of $\mathbf{w}$ can be approached in different ways [DH73], but it is often replaced by the problem of minimizing a Mean Square Error (MSE) cost function, which differs from (2) in that the nonlinear function $\theta_0$ has been removed.

## 2.2   CURVATURE PROPERTIES OF THE $MSE$ COST FUNCTION

The three structures that we investigate rely on curvature properties of the $MSE$ cost function. Consider the dependence of $MSE$ on one of the parameters $w_i$. Training leads to the optimal value $w_i^*$ for this parameter. One way of reducing the capacity is to set $w_i$ to zero. For the linear classifier, this reduces the VC-dimension by one: $h' = dim(\mathbf{w}) - 1 = n - 1$. The $MSE$ increase resulting from setting $w_i = 0$ is to lowest order proportional to the curvature of the $MSE$ at $w_i^*$. Since the decrease in capacity should be achieved at the smallest possible expense in $MSE$ increase, directions in weight space corresponding to small $MSE$ curvature are good candidates for elimination.

The curvature of the $MSE$ is specified by the Hessian matrix $H$ of second derivatives of the $MSE$ with respect to the weights. For a linear classifier, the Hessian matrix is given by twice the correlation matrix of the training inputs, $H = (2/p) \sum_{k=1}^{p} \mathbf{x}^k \mathbf{x}^{kT}$. The Hessian matrix is symmetric, and can be diagonalized to get rid of cross terms,

to facilitate decisions about the simultaneous elimination of several directions in weight space. The elements of the Hessian matrix after diagonalization are the eigenvalues $\lambda_i$; the corresponding eigenvectors give the principal directions $w_i'$ of the $MSE$. In the rotated axis, the increase $\Delta MSE$ due to setting $w_i' = 0$ takes a simple form:

$$\Delta MSE \approx \frac{1}{2} \lambda_i (w_i'^*)^2 \qquad (3)$$

The quadratic approximation becomes an exact equality for the linear classifier. Principal directions $w_i'$ corresponding to small eigenvalues $\lambda_i$ of $H$ are good candidates for elimination.

## 2.3   PRINCIPAL COMPONENT ANALYSIS

One common way of reducing the capacity of a classifier is to reduce the dimension of the input space and thereby reduce the number of necessary free parameters (or weights). *Principal Component Analysis* (PCA) is a feature extraction method based on eigenvalue analysis. Input vectors **x** of dimension $n$ are approximated by a linear combination of $m \leq n$ vectors forming an ortho-normal basis. The coefficients of this linear combination form a vector **x**′ of dimension $m$. The optimal basis in the least square sense is given by the $m$ eigenvectors corresponding to the $m$ largest eigenvalues of the correlation matrix of the training inputs (this matrix is $1/2$ of $H$). A structure is obtained by ranking the classifiers according to $m$. The VC-dimension of the classifier is reduced to: $h' = dim(\mathbf{x}') = m$.

## 2.4   OPTIMAL BRAIN DAMAGE

For a linear classifier, pruning can be implemented in two different but equivalent ways: (i) change input coordinates to a principal axis representation, prune the *components* corresponding to small eigenvalues according to PCA, and then train with the $MSE$ cost function; (ii) change coordinates to a principal axis representation, train with $MSE$ first, and then prune the *weights*, to get a weight vector **w**′ of dimension $m \leq n$. Procedure (i) can be understood as a preprocessing, whereas procedure (ii) involves an *a posteriori* modification of the structure of the classifier (network architecture). The two procedures become identical if the weight elimination in (ii) is based on a 'smallest eigenvalue' criterion.

Procedure (ii) is very reminiscent of *Optimal Brain Damage* (OBD), a weight pruning procedure applied after training. In OBD, the best candidates for pruning are those weights which minimize the increase $\Delta MSE$ defined in equation (3). The $m$ weights that are kept do not necessarily correspond to the largest $m$ eigenvalues, due to the extra factor of $(w_i'^*)^2$ in equation (3). In either implementation, the VC-dimension is reduced to $h' = dim(\mathbf{w}') = dim(\mathbf{x}') = m$.

## 2.5   WEIGHT DECAY

Capacity can also be controlled through an additional term in the cost function, to be minimized simultaneously with $MSE$. Linear classifiers can be ranked according to the norm $\|\mathbf{w}\|^2 = \sum_{j=1}^{n} w_j^2$ of the weight vector. A structure is constructed

by allowing within the subset $S_r$ only those classifiers which satisfy $\|\mathbf{w}\|^2 \leq c_r$. The positive bounds $c_r$ form an increasing sequence: $c_1 < c_2 < ... < c_r < ...$ This sequence can be matched with a monotonically decreasing sequence of positive Lagrange multipliers $\gamma_1 \geq \gamma_2 \geq ... \geq \gamma_r \geq ...$, such that our training problem stated as the minimization of $MSE$ within a specific set $S_r$ is implemented through the minimization of a new cost function: $MSE + \gamma_r\|\mathbf{w}\|^2$. This is equivalent to the Weight Decay procedure (WD). In a mechanical analogy, the term $\gamma_r\|\mathbf{w}\|^2$ is like the energy of a spring of tension $\gamma_r$ which pulls the weights to zero. As it is easier to pull in the directions of small curvature of the $MSE$, WD pulls the weights to zero predominantly along the principal directions of the Hessian matrix $H$ associated with small eigenvalues.

In the principal axis representation, the minimum $\mathbf{w}^\gamma$ of the cost function $MSE + \gamma\|\mathbf{w}\|^2$, is a simple function of the minimum $\mathbf{w}^0$ of the $MSE$ in the $\gamma \rightarrow 0^+$ limit: $w_i^\gamma = w_i^0 \lambda_i/(\lambda_i + \gamma)$. The weight $w_i^0$ is attenuated by a factor $\lambda_i/(\lambda_i + \gamma)$. Weights become negligible for $\gamma \gg \lambda_i$, and remain unchanged for $\gamma \ll \lambda_i$. The effect of this attenuation can be compared to that of weight pruning. Pruning all weights such that $\lambda_i < \gamma$ reduces the capacity to:

$$h' = \sum_{i=1}^{n} \theta_\gamma(\lambda_i) \,, \tag{4}$$

where $\theta_\gamma(u) = 1$ if $u > \gamma$ and $\theta_\gamma(u) = 0$ otherwise.

By analogy, we introduce the Weight Decay capacity:

$$h' = \sum_{i=1}^{n} \frac{\lambda_i}{\lambda_i + \gamma} \,. \tag{5}$$

This expression arises in various theoretical frameworks [Moo92,McK92], and is valid only for broad spectra of eigenvalues.

# 3   SMOOTHING, HIGHER-ORDER UNITS, AND REGULARIZATION

Combining several different structures achieves further performance improvements. The combination of exponential smoothing (a preprocessing transformation of input space) and regularization (a modification of the learning algorithm) is shown here to improve character recognition. The generalization ability is dramatically improved by the further introduction of second-order units (an architectural modification).

## 3.1   SMOOTHING

Smoothing is a preprocessing which aims at reducing the effective dimension of input space by degrading the resolution: after smoothing, decimation of the inputs could be performed without further image degradation. Smoothing is achieved here through convolution with an exponential kernel:

$$BLURRED.PIXEL(i,j) = \frac{\sum_k \sum_l PIXEL(i+k,j+l) \, exp[-\frac{1}{\beta}\sqrt{k^2+l^2}]}{\sum_k \sum_l exp[-\frac{1}{\beta}\sqrt{k^2+l^2}]},$$

where $\beta$ is the smoothing parameter which determines the structure.

Convolution with the chosen kernel is an invertible linear operation. Such preprocessing results in no capacity change for a MSE-trained linear classifier. Smoothing only modifies the spectrum of eigenvalues and must be combined with an eigenvalue-based regularization procedure such as OBD or WD, to obtain performance improvement through capacity decrease.

## 3.2 HIGHER-ORDER UNITS

Higher-order (or sigma-pi) units can be substituted for the linear units to get polynomial classifiers: $F(\mathbf{x}, \mathbf{w}) = \theta_0(\mathbf{w}^T \xi(\mathbf{x}))$, where $\xi(\mathbf{x})$ is an m-dimensional vector $(m \geq n)$ with components: $x_1, x_2, ..., x_n, (x_1 x_1), (x_1 x_2), ..., (x_n x_n), ..., (x_1 x_2 ... x_n)$. The structure is geared towards increasing the capacity, and is controlled by the order of the polynomial: $S_1$ contains all the linear terms, $S_2$ linear plus quadratic, etc. Computations are kept tractable with the method proposed in reference [Pog75].

# 4   EXPERIMENTAL RESULTS

Experiments were performed on the benchmark problem of handwritten digit recognition described in reference [GPP$^+$89]. The database consists of 1200 ($16 \times 16$) binary pixel images, divided into 600 training examples and 600 test examples.

In figure 1, we compare the results obtained by pruning inputs or weights with PCA and the results obtained with WD. The overall appearance of the curves is very similar. In both cases, the capacity (computed from (4) and (5)) decreases as a function of $\gamma$, whereas the training error increases. For the optimum value $\gamma*$, the capacity is only 1/3 of the nominal capacity, computed solely on the basis of the network architecture. At the price of some error on the training set, the error rate on the test set is only half the error rate obtained with $\gamma = 0^+$.

The competition between capacity and training error always results in a unique minimum of the guaranteed risk (1). It is remarkable that our experiments show the minimum of $E_{guarant}$ coinciding with the minimum of $E_{test}$. Any of these two quantities can therefore be used to determine $\gamma*$. In principle, another independent test set should be used to get a reliable estimate of $E_{gene}$ (cross-validation). It seems therefore advantageous to determine $\gamma*$ using the minimum of $E_{guarant}$ and use the test set to predict the generalization performance.

Using $E_{guarant}$ to determine $\gamma*$ raises the problem of determining the capacity of the system. The capacity can be measured when analytic computation is not possible. Measurements performed with the method proposed by Vapnik, Levin, and Le Cun yield results in good agreement with those obtained using (5). The method yields an *effective VC-dimension* which accounts for the global capacity of the system, including the effects of input data, architecture, and learning algorithm [2].

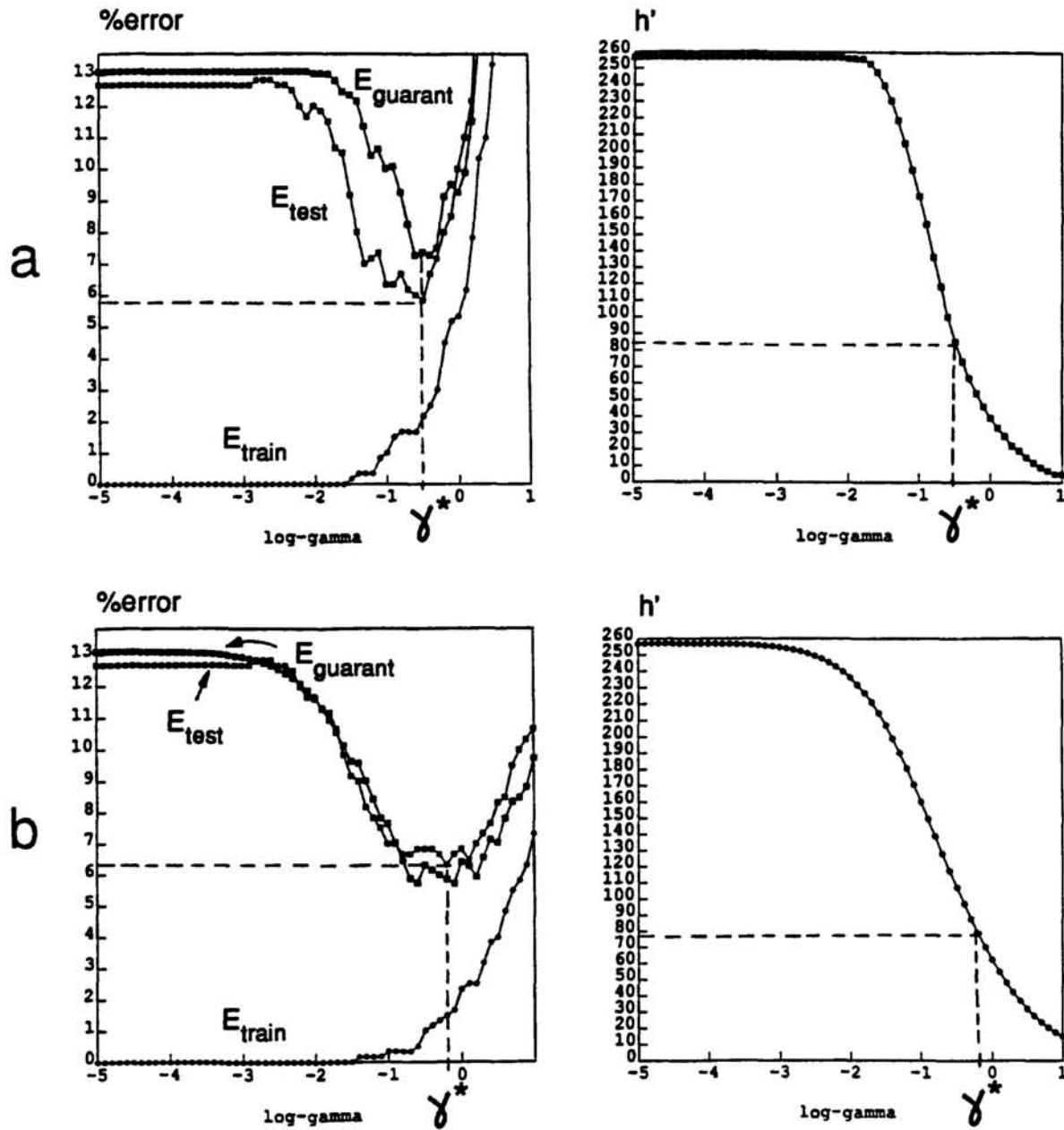

Figure 1: Percent error and capacity $h'$ as a function of $\log \gamma$ (linear classifier, no smoothing): (a) weight/input pruning via PCA ($\gamma$ is a threshold), (b) WD ($\gamma$ is the decay parameter). The guaranteed risk has been rescaled to fit in the figure.

Table 1: $E_{test}$ for Smoothing, WD, and Higher-Order Combined.

| $\beta$ | $\gamma$ | $1^{st}$ order | $2^{nd}$ order |
|---------|----------|---------------|----------------|
| 0 | $\gamma^*$ | 6.3 | 1.5 |
| 1 | $\gamma^*$ | 5.0 | 0.8 |
| 2 | $\gamma^*$ | 4.5 | 1.2 |
| 10 | $\gamma^*$ | 4.3 | 1.3 |
| any | $0^+$ | 12.7 | 3.3 |

In table 1 we report results obtained when several structures are combined. Weight decay with $\gamma = \gamma^*$ reduces $E_{test}$ by a factor of 2. Input space smoothing used in conjunction with WD results in an additional reduction by a factor of 1.5. The best performance is achieved for the highest level of smoothing, $\beta = 10$, for which the blurring is considerable. As expected, smoothing has no effect in the absence of WD.

The use of second-order units provides an additional factor of 5 reduction in $E_{test}$. For second order units, the number of weights scales like the square of the number of inputs $n^2 = 66049$. But the capacity (5) is found to be only 196, for the optimum values of $\gamma$ and $\beta$.

## 5    CONCLUSIONS AND EPILOGUE

Our results indicate that the VC-dimension must measure the global capacity of the system. It is crucial to incorporate the effects of preprocessing of the input data and modifications of the learning algorithm. Capacities defined solely on the basis of the network architecture give overly pessimistic upper bounds.

The method of SRM provides a powerful tool for tuning the capacity. We have shown that structures acting at different levels (preprocessing, architecture, learning mechanism) can produce similar effects. We have then combined three different structures to improve generalization. These structures have interesting complementary properties. The introduction of higher-order units increases the capacity. Smoothing and weight decay act in conjunction to decrease it.

Elaborate neural networks for character recognition [LBD+90,GAL+91] also incorporate similar complementary structures. In multilayer sigmoid-unit networks, the capacity is increased through additional hidden units. Feature extracting neurons introduce smoothing, and regularization follows from prematurely stopping training before reaching the $MSE$ minimum. When initial weights are chosen to be small, this stopping technique produces effects similar to those of weight decay.

## Acknowledgments

We wish to thank L. Jackel's group at Bell Labs for useful discussions, and are particularly grateful to E. Levin and Y. Le Cun for communicating to us the unpublished method of computing the effective VC-dimension.

## Footnotes

[1]We assume, for simplicity, that the first component of vector $\mathbf{x}$ is constant and set to 1, so that the corresponding weight introduces the bias value.

[2]Schematically, measurements of the *effective VC-dimension* consist of splitting the training data into two subsets. The difference between $E_{train}$ in these subsets is maximized. The value of $h$ is extracted from the fit to a theoretical prediction for such maximal discrepancy.

## References

[DH73]     R.O. Duda and P.E. Hart. *Pattern Classification And Scene Analysis.* Wiley and Son, 1973.

[GAL+91]  I. Guyon, P. Albrecht, Y. Le Cun, J. Denker, and W. Hubbard. Design of a neural network character recognizer for a touch terminal. *Pattern Recognition*, 24(2), 1991.

[GPP+89]  I. Guyon, I. Poujaud, L. Personnaz, G. Dreyfus, J. Denker, and Y. Le Cun. Comparing different neural network architectures for classifying handwritten digits. In *Proceedings of the International Joint Conference on Neural Networks*, volume II, pages 127–132. IEEE, 1989.

[LBD+90]  Y. Le Cun, B. Boser, J. S. Denker, D. Henderson, R. E. Howard, W. Hubbard, and L. D. Jackel. Back-propagation applied to handwritten zipcode recognition. *Neural Computation*, 1(4), 1990.

[LDS90]    Y. Le Cun, J. S. Denker, and S. A. Solla. Optimal brain damage. In D. S. Touretzky, editor, *Advances in Neural Information Processing Systems 2 (NIPS 89)*, pages 598–605. Morgan Kaufmann, 1990.

[McK92]    D. McKay. A practical bayesian framework for backprop networks. In *this volume*, 1992.

[Moo92]    J. Moody. Generalization, weight decay and architecture selection for non-linear learning systems. In *this volume*, 1992.

[Pog75]     T. Poggio. On optimal nonlinear associative recall. *Biol. Cybern.*, (9)201, 1975.

[The89]     C. W. Therrien. *Decision, Estimation and Classification: An Introduction to Pattern Recognition and Related Topics.* Wiley, 1989.

[Vap82]     V. Vapnik. *Estimation of Dependences Based on Empirical Data.* Springer-Verlag, 1982.

[Vap92]     V Vapnik. Principles of risk minimization for learning theory. In *this volume*, 1992.